# Evidence for a Forward Dynamics Model in Human Adaptive Motor Control

**Nikhil Bhushan and Reza Shadmehr**
Dept. of Biomedical Engineering
Johns Hopkins University, Baltimore, MD 21205
Email: nbhushan@bme.jhu.edu, reza@bme.jhu.edu

## Abstract

Based on computational principles, the concept of an internal model for adaptive control has been divided into a forward and an inverse model. However, there is as yet little evidence that learning control by the CNS is through adaptation of one or the other. Here we examine two adaptive control architectures, one based only on the inverse model and other based on a combination of forward and inverse models. We then show that for reaching movements of the hand in novel force fields, only the learning of the forward model results in key characteristics of performance that match the kinematics of human subjects. In contrast, the adaptive control system that relies only on the inverse model fails to produce the kinematic patterns observed in the subjects, despite the fact that it is more stable. Our results provide evidence that learning control of novel dynamics is via formation of a forward model.

## 1 Introduction

The concept of an internal model, a system for predicting behavior of a controlled process, is central to the current theories of motor control (Wolpert et al. 1995) and learning (Shadmehr and Mussa-Ivaldi 1994). Theoretical studies have proposed that internal models may be divided into two varieties: forward models, which simulate the causal flow of a process by predicting its state transition given a motor command, and inverse models, which estimate motor commands appropriate for a desired state transition (Miall and Wolpert, 1996). This classification is relevant for adaptive control because based on computational principles, it has been proposed that learning control of a nonlinear system might be facilitated if a forward model of the plant is learned initially, and then during an off-line period is used to train an inverse model (Jordan and Rumelhart, 1992). While there is no experimental evidence for this idea in the central nervous system, there is substantial evidence

that learning control of arm movements involves formation of an internal model. For example, practicing arm movements while holding a novel dynamical system initiates an adaptation process which results in the formation of an internal model: upon sudden removal of the force field, after-effects are observed which match the expected behavior of a system that has learned to predict and compensate for the dynamics of the imposed field (Shadmehr and Brashers-Krug, 1997). However, the computational nature of this internal model, whether it be a forward or an inverse model, or a combination of both, is not known.

Here we use a computational approach to examine two adaptive control architectures: adaptive inverse model feedforward control and adaptive forward-inverse model feedback control. We show that the two systems predict different behaviors when applied to control of arm movements. While adaptation to a force field is possible with either approach, the second system with feedback control through an adaptive forward model, is far less stable and is accompanied with distinct kinematic signatures, termed "near path-discontinuities". We observe remarkably similar instability and near path-discontinuities in the kinematics of 16 subjects that learned force fields. This is behavioral evidence that learning control of novel dynamics is accomplished with an adaptive forward model of the system.

## 2 Adaptive Control using Internal Models

Adaptive control of a nonlinear system which has large sensory feedback delays, such as the human arm, can be accomplished by using two different internal model architectures. The first method uses only an adaptive inverse dynamics model to control the system (Shadmehr and Mussa-Ivaldi, 1994). The adaptive controller is feedforward in nature and ignores delayed feedback during the movement. The control system is stable because it relies on the equilibrium properties of the muscle and the spinal reflexes to correct for any deviations from the desired trajectory. The second method uses a rapidly adapting forward dynamics model and delayed sensory feedback in addition to an inverse dynamics model to control arm movements (Miall and Wolpert, 1996). In this case, the corrections to deviations from the desired trajectory are a result of a combination of supraspinal feedback as well as spinal/muscular feedback. Since the two methods rely on different internal model and feedback structures, they are expected to behave differently when the dynamics of the system are altered.

### The Mechanical Model of the Human Arm

For the purpose of simulating arm movements with the two different control architectures, a reasonably accurate model of the human arm is required. We model the arm as a two joint revolute arm attached to six muscles that act in pairs around the two joints. The three muscle pairs correspond to elbow joint, shoulder joint and two joint muscles and are assumed to have constant moment arms. Each muscle is modeled using a Hill parametric model with nonlinear stiffness and viscosity (Soechting and Flanders, 1997). The dynamics of the muscle can be represented by a nonlinear state function $f_M$, such that,

$$F_t = f_M(N, x_m, \dot{x}_m) \tag{1}$$

where, $F_t$ is the force developed by the muscle, $N$ is the neural activation to the muscle, and $x_m$, $\dot{x}_m$ are the muscle length and velocity. The passive dynamics related to the mechanics of the two-joint revolute arm can be represented by $f_D$, such that,

$$\ddot{x} = f_D(T, x, \dot{x}) = D^{-1}(x)[T - C(x, \dot{x})\dot{x} + J^T F_x] \tag{2}$$

where, $\ddot{x}$ is the hand acceleration, $T$ is the joint torque generated by the muscles, $x$, $\dot{x}$ are the hand position and velocity, $D$ and $C$ are the inertia and the coriolis matrices of the arm, $J$ is the Jacobian for hand position and joint angle, and $F_x$ is the external dynamic interaction force on the hand.

Under the force field environment, the external force $F_x$ acting on the hand is equal to $B\dot{x}$, where $B$ is a 2x2 rotational viscosity matrix. The effect of the force field is to push the hand perpendicular to the direction of movement with a force proportional to the speed of the hand. The overall forward plant dynamics of the arm is a combination of $f_M$ and $f_D$ and can be represented by the function $f_p$,

$$\ddot{x} = f_p(N, x, \dot{x}) \tag{3}$$

## Adaptive Inverse Model Feedforward Control

The first control architecture uses a feedforward controller with only an adaptive inverse model. The inverse model computes the neural activation to the muscles for achieving a desired acceleration, velocity and position of the hand. It can be represented as the estimated inverse, $\hat{f}_p^{-1}$, of the forward plant dynamics, and maps the desired position $x_d$, velocity $\dot{x}_d$, and acceleration $\ddot{x}_d$ of the hand, into descending neural commands $N_C$.

$$N_C = \hat{f}_p^{-1}(\ddot{x}_d, x_d, \dot{x}_d) \tag{4}$$

Adaptation to novel external dynamics occurs by learning a new inverse model of the altered external environment. The error between desired and actual hand trajectory can be used for training the inverse model. When the inverse model is an exact inverse of the forward plant dynamics, the gain of the feedforward path is unity and the arm exactly tracks the desired trajectory. Deviations from the desired trajectory occur when the inverse model does not exactly model the external dynamics. Under that situation, the spinal reflex corrects for errors in desired $(x_{md}, \dot{x}_{md})$ and actual $(x_m, \dot{x}_m)$ muscle state, by producing a corrective neural signal $N_R$ based on a linear feedback controller with constants $K_1$ and $K_2$.

$$N_R = K_1(x_{md} - x_m) + K_2(\dot{x}_{md} - \dot{x}_m) \tag{5}$$

## Adaptive Forward-Inverse Model Feedback Control

The second architecture provides feedback control of arm movements in addition to the feedforward control described above. Delays in feedback cause instability, therefore, the system relies on a forward model to generate updated state estimates of the arm. An estimated error in hand trajectory is given by the difference in desired and estimated state, and can be used by the brain to issue corrective neural signals to the muscles while a movement is being made. The forward model, written

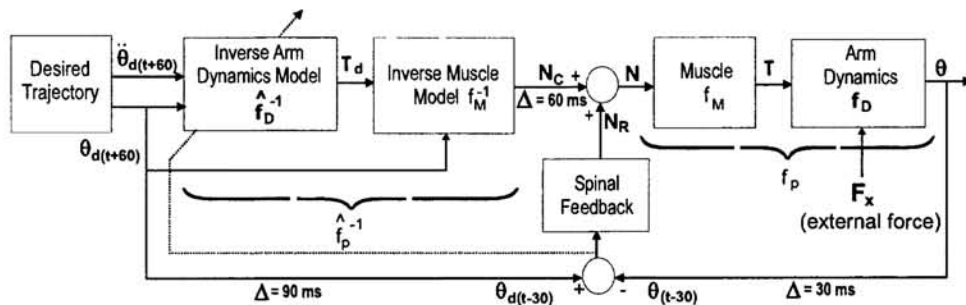

Figure 1: The adaptive inverse model feedforward control system.

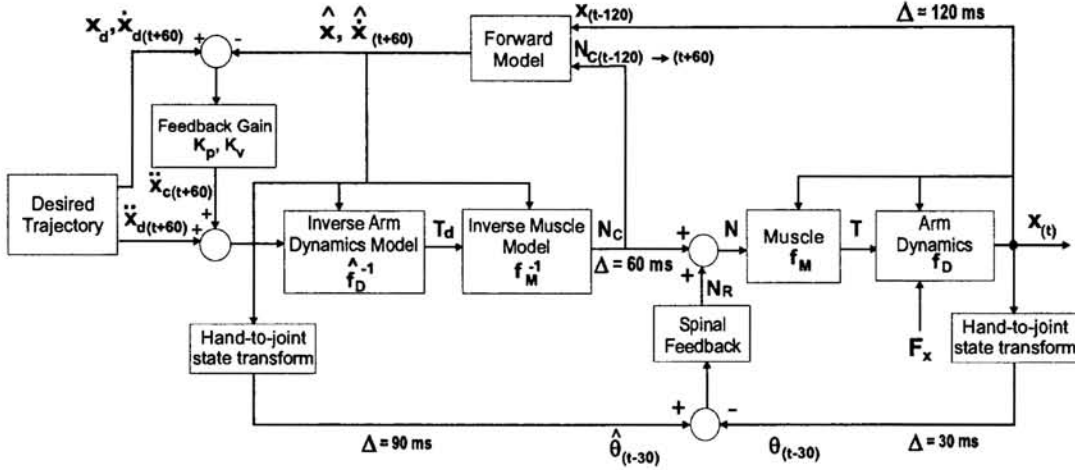

Figure 2: A control system that provides feedback control with the use of a forward and an inverse model.

as $\hat{f}_p$, mimics the forward dynamics of the plant and predicts hand acceleration $\hat{\ddot{x}}$, from neural signal $N_C$, and an estimate of hand state $\hat{x}, \hat{\dot{x}}$.

$$\hat{\ddot{x}} = \hat{f}_p(N_C, \hat{x}, \hat{\dot{x}}) \tag{6}$$

Using this equation, one can solve for $\hat{x}, \hat{\dot{x}}$ at time $t$, when given the estimated state at some earlier time $t - \tau$, and the descending neural commands $N_C$ from time $t - \tau$ to $t$. If $t$ is the current time and $\tau$ is the time delay in the feedback loop, then sensory feedback gives the hand state $x, \dot{x}$ at $t - \tau$. The current estimate of the hand position and velocity can be computed by assuming initial conditions $\hat{x}(t - \tau) = x(t - \tau)$ and $\hat{\dot{x}}(t - \tau) = \dot{x}(t - \tau)$, and then solving Eq. 6. For the simulations, $\tau$ has value of 200 msec, and is composed of 120 msec feedback delay, 60 msec descending neural path delay, and 20 msec muscle activation delay.

Based on the current state estimate and the estimated error in trajectory, the desired acceleration is corrected using a linear feedback controller with constants $K_p$ and $K_v$. The inverse model maps the hand acceleration to appropriate neural signal for the muscles $N_C$. The spinal reflex provides additional corrective feedback $N_R$, when there is an error in the estimated and actual muscle state.

$$\ddot{x}_{new} = \ddot{x}_d + \ddot{x}_c = \ddot{x}_d + K_p(x_d - \hat{x}) + K_v(\dot{x}_d - \hat{\dot{x}}) \tag{7}$$

$$N_C = \hat{f}_p^{-1}(\ddot{x}_{new}, \hat{x}, \hat{\dot{x}}) \tag{8}$$

$$N_R = K_1(\hat{x}_m - x_m) + K_2(\hat{\dot{x}}_m d - \dot{x}_m) \tag{9}$$

When the forward model is an exact copy of the forward plant dynamics $\hat{f}_p = f_p$, and the inverse model is correct $\hat{f}_p^{-1} = f_p^{-1}$, the hand exactly tracks the desired trajectory. Errors due to an incorrect inverse model are corrected through the feedback loop. However, errors in the forward model cause deviations from the desired behavior and instability in the system due to inappropriate feedback action.

## 3 Simulations results and comparison to human behavior

To test the two control architectures, we compared simulations of arm movements for the two methods to experimental human results under a novel force field environment. Sixteen human subjects were trained to make rapid point-to-point reaching

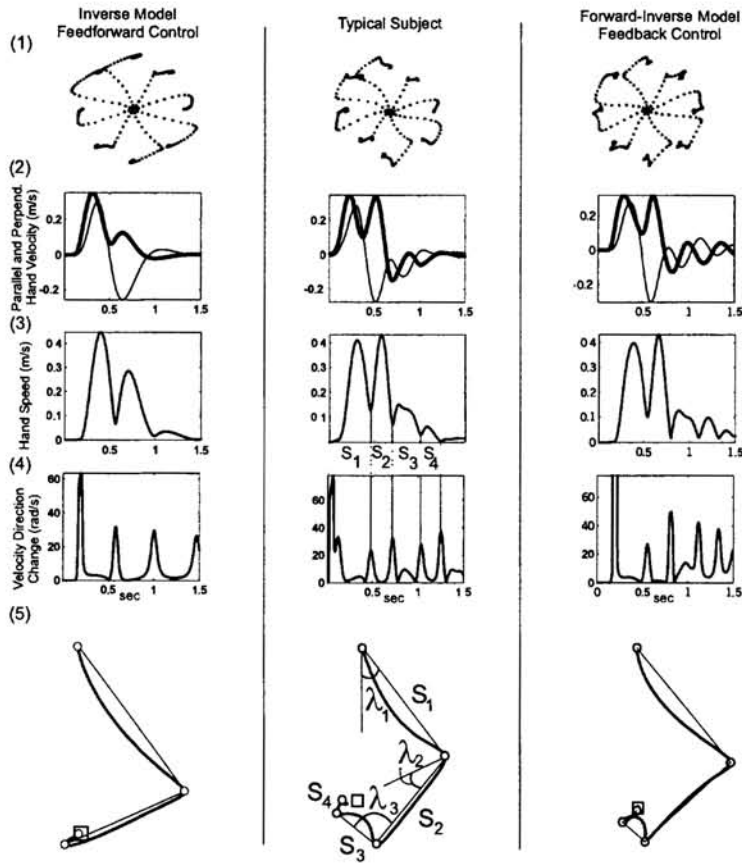

Figure 3: Performance in field $B_2$ after a typical subject (middle column) and each of the controllers (left and right columns) had adapted to field $B_1$. (1) hand paths for 8 movement directions, (2-5) hand velocity, speed, derivative of velocity direction, and segmented hand path for the $-90^o$ downward movement. The segmentation in hand trajectory that is observed in our subjects is almost precisely reproduced by the controller that uses a forward model.

movements with their hand while an external force field, $F_x = B\dot{x}$, pushed on the hand. The task was to move the hand to a target position 10 cm away in 0.5 sec. The movement could be directed in any of eight equally spaced directions. The subjects made straight-path minimum-jerk movements to the targets in the absence of any force fields. The subjects were initially trained in force field $B_1$ with $B$=[0 13;-13 0], until they had completely adapted to this field and converged to the straight-path minimum-jerk movement observed before the force field was applied. Subsequently, the force field was switched to $B_2$ with $B$=[0 -13;13 0] (the new field pushed anticlockwise, instead of clockwise), and the first three movements in each direction were used for data analysis. The movements of the subjects in field $B_2$ showed huge deviations from the desired straight path behavior because the subjects expected clockwise force field $B_1$. The hand trajectories for the first movement in each of the eight directions are shown for a typical subject in Fig. 3 (middle column).

Simulations were performed for the two methods under the same conditions as the human experiment. The movements were made in force field $B_2$, while the internal models were assumed to be adapted to field $B_1$. Complete adaptation to the force field $B_1$ was found to occur for the two methods only when both

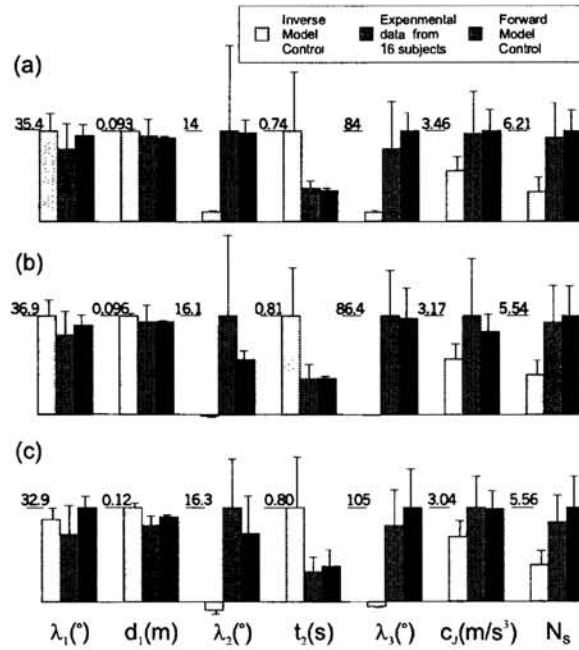

Figure 4: The mean and standard deviation for segmentation parameters for each type of controller as compared to the data from our subjects. Parameters are defined in Fig. 3: $\lambda_i$ is angle about a seg. point, $d_i$ is the distance to the $i$-th seg. point, $t_i$ is time to reach the $i$-th seg. point, $c_j$ is cumulative squared jerk for the entire movement, $N_s$ is number of seg. point in the movement. Up until the first segmentation point ($\lambda_1$ and $d_1$), behavior of the controllers are similar and both agree with the performance of our subjects. However, as the movement progresses, only the controller that utilizes a forward model continues to agree with the movement characteristics of the subjects.

the inverse and forward models expected field $B_1$. Fig. 3 (left column) shows the simulation of the adaptive inverse model feedforward control for movements in field $B_2$ with the inverse model incorrectly expecting $B_1$. Fig. 3 (right column) shows the simulation of the adaptive forward-inverse model feedback control for movements in field $B_2$ with both the forward and the inverse model incorrectly expecting $B_1$. Simulations with the two methods show clear differences in stability and corrective behavior for all eight directions of movement. The simulations with the inverse model feedforward control seem to be stable, and converge to the target along a straight line after the initial deviation. The simulations with the forward-inverse model feedback control are more unstable and have a curious kinematic pattern with discontinuities in the hand path. This is especially marked for the downward movement. The subject's hand paths show the same kinematic pattern of near discontinuities and segmentation of movement as found with the forward-inverse model feedback control.

To quantify the segmentation pattern in the hand path, we identified the "near path-discontinuities" as points on the trajectory where there was a sudden change in both the derivative of hand speed and the direction of hand velocity. The hand path was segmented on the basis of these near discontinuities. Based on the first three segments in the hand trajectory we defined the following parameters: $\lambda_1$, angle between the first segment and the straight path to the target; $d_1$, the distance covered during the first segment; $\lambda_2$, angle between the second segment and straight path to the target from the first segmentation point; $t_2$, time duration of the second

segment; $\lambda_3$, angle between the second and third segments; $N_S$, the number of segmentation points in the movement. We also calculated the cumulative jerk $C_J$ in the movements to get a measure of the instability in the system.

The results of the movement segmentation are presented in Fig. 4 for 16 human subjects, 25 simulations of the inverse model and 20 simulations of the forward model control for three movement directions (a) $-90^\circ$ downward, (b) $90^\circ$ upward and (c) $135^\circ$ upward. We performed the different simulations for the two methods by systematically varying various model parameters over a reasonable physiological range. This was done because the parameters are only approximately known and also vary from subject to subject. The parameters of the second and third segment, as represented by $\lambda_2$, $t_2$ and $\lambda_3$, clearly show that the forward model feedback control performs very differently from inverse model feedforward control and the behavior of human subjects is very well predicted by the former. Furthermore, this characteristic behavior could be produced by the forward-inverse model feedback control only when the forward model expected field $B_1$. This could be accomplished only by adaptation of the forward model during initial practice in field $B_1$. This provides evidence for an adaptive forward model in the control of human arm movements in novel dynamic environments.

We further tried to fit adaptation curves of simulated movement parameters (using forward-inverse model feedback control) to real data as subjects trained in field $B_1$. We found that the best fit was obtained for a rapidly adapting forward and inverse model (Bhushan and Shadmehr, 1999). This eliminated the possibility that the inverse model was trained offline after practice. The data, however, suggested that during learning of a force field, the rate of learning of the forward model was faster than the inverse model. This finding could be pariculary relevant if it is proven that a forward model is easier to learn than an inverse model (Narendra, 1990), and could provide a computational rationale for the existence of forward model in adaptive motor control.

## References

Bhushan N, Shadmehr R (1999) Computational architecture of the adaptive controller during learning of reaching movements in force fields. *Biol Cybern*, in press.

Jordan MI, Flash T, Arnon Y (1994) A model of learning arm trajectories from spatial deviations *Journal of Cog Neur* 6:359-376.

Jordan MI, Rumelhart DE (1992) Forward model: supervised learning with a distal teacher. *Cog Sc* 16:307-354.

Miall RC, Wolpert DM (1996) Forward models for physiological motor control. *Neural Networks* 9:1265-1279.

Narendra KS (1990) Identification and control of dynamical systems using neural networks. *Neural Networks* 1:4-27.

Shadmehr R, Brashers-Krug T (1997) Functional stages in the formation of human long-term memory. *J Neurosci* 17:409-19.

Shadmehr R, Mussa-Ivaldi FA (1994) Adaptive representation of dynamics during learning of a motor task. *The Journal of Neuroscience* 14:3208-3224.

Soechting JF, Flanders M (1997) Evaluating an integrated musculoskeletal model of the human arm *J Biomech Eng* 9:93-102.

Wolpert DM, Ghahramani Z, Jordan MI (1995) An internal model for sensorimotor integration. *Science* 269:1880-82.
